# Sequence learning with hidden units in spiking neural networks

**Johanni Brea, Walter Senn and Jean-Pascal Pfister**
Department of Physiology
University of Bern
Bühlplatz 5
CH-3012 Bern, Switzerland
`{brea, senn, pfister}@pyl.unibe.ch`

## Abstract

We consider a statistical framework in which recurrent networks of spiking neurons learn to generate spatio-temporal spike patterns. Given biologically realistic stochastic neuronal dynamics we derive a tractable learning rule for the synaptic weights towards hidden and visible neurons that leads to optimal recall of the training sequences. We show that learning synaptic weights towards hidden neurons significantly improves the storing capacity of the network. Furthermore, we derive an approximate online learning rule and show that our learning rule is consistent with Spike-Timing Dependent Plasticity in that if a presynaptic spike shortly precedes a postsynaptic spike, potentiation is induced and otherwise depression is elicited.

## 1 Introduction

Learning to produce temporal sequences is a general problem that the brain needs to solve. Movements, songs or speech, all require the generation of specific spatio-temporal patterns of neural activity that have to be learned. Early attempts to model sequence learning used a simple asymmetric Hebbian learning rule [10, 20, 6] and succeeded to store sequences of random patterns, but perform poorly as soon as there are temporal correlations between the patterns [3].

Later work on pattern storage or sequence learning recognized the need for matching the storage rule with the recall dynamics [2, 18, 12] and derived the optimal storage rule for a given recall dynamics [2, 18] or an optimal recall dynamics for a given storage rule [12], but didn't consider hidden neurons and therefore restricted the class of possible patterns to be learned. Other studies [14] included a reservoir of hidden neurons but assumed weights towards the hidden neurons to be fixed. Finally, Boltzmann machines [1] - which learn to produce a given distribution of patterns with visible and hidden neurons - applied to sequence learning [9, 22, 21] are trained with Contrastive Divergence [8] and either an approximation that neglects the influence of the future or use a non-local and non-causal learning rule.

Here we start by defining a stochastic neuronal dynamics - that can be arbitrarily complicated (e.g. with non-Markovian dependencies). This stochastic dynamics defines the overall probability distribution which is parametrized by the synaptic weights. The goal of learning is to adapt synaptic weights such that the model distribution approximates as good as possible the target distribution of temporal sequences. This can be seen as the extension of the maximum likelihood approach of Barber [2] where we add stochastic hidden neurons with plastic weights. In order to learn the weights, we implement a variant of the Expectation-Maximization (EM) algorithm [5] where we use importance sampling in the expectation step in a way that makes the sampling procedure easy.

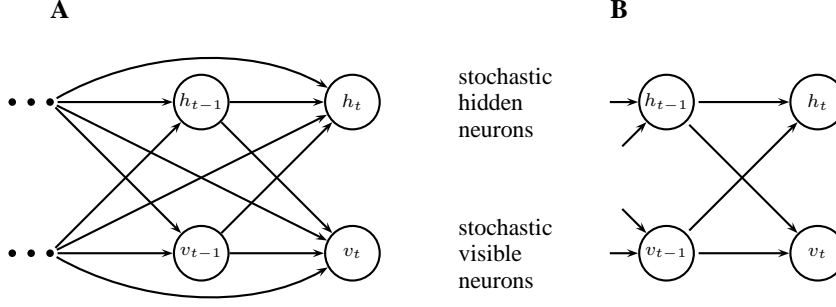

Figure 1: Graphical representation of the conditional dependencies of the joint distribution over visible and hidden sequences. **A** Graphical model used for the derivation of the learning rule in section 2 and the example in section 4. **B** Markovian model used in the example with binary neurons in section 3.

The resulting learning rule is local (but modulated by a global factor), causal and biologically relevant in the sense that it shares important features with Spike-Timing Dependent Plasticity (STDP). We also derive an online version of the learning rule and show numerically that it performs almost equally well as the exact batch learning rule.

## 2 Learning a distribution of sequences

Let us consider temporal sequences $v = \{v_{t,i} | t = 0 \ldots T, i = 1 \ldots N_v\}$ of $N_v$ visible neurons over the interval $[0, T]$. We will use the notation $v_t = \{v_{t,i} | i = 1 \ldots N_v\}$ and $v_{t_1:t_2} = \{v_{t,i} | t = t_1 \ldots t_2, i = 1 \ldots N_v\}$ to denote parts of the sequence. Note that $v = v_{0:T}$ denotes the whole sequence. Those visible sequences $v$ are drawn i.i.d. from a target distribution $P^*(v)$ that must be learned by a model which consists of $N_v$ visible neurons and $N_h$ hidden neurons. The model distribution over those visible sequences is denoted by $P_\theta(v) = \sum_h P_\theta(v, h)$ where $\theta$ denotes the model parameters, $h = \{h_{t,i} | t = 0 \ldots T, i = 1 \ldots N_h\}$ the hidden temporal sequence and $P_\theta(v, h)$ the joint distribution over the visible and the hidden sequences. The natural way to quantify the mismatch between the target distribution $P^*(v)$ and the model distribution $P_\theta(v)$ is given by the Kullback-Leibler divergence:

$$D_{\mathrm{KL}}(P^*(v) || P_\theta(v)) = \sum_v P^*(v) \log \frac{P^*(v)}{P_\theta(v)} . \tag{1}$$

If the joint model distribution $P_\theta(v, h)$ is differentiable with respect to the model parameters $\theta$, then the sequence learning problem can be phrased as gradient descent on the KL divergence in Eq. (1):

$$\Delta \theta = \eta \left\langle \frac{\partial \log P_\theta(v, h)}{\partial \theta} \right\rangle_{P_\theta(h|v) P^*(v)} , \tag{2}$$

where $\eta$ is the learning rate and we used the fact that $\frac{\partial}{\partial \theta} \log P_\theta(v) = \frac{1}{P_\theta(v)} \frac{\partial}{\partial \theta} \sum_h P_\theta(v, h) = \sum_h P_\theta(h|v) \frac{\partial}{\partial \theta} \log P_\theta(v, h)$. Eq. (2) can be seen as a variant of the EM algorithm [5, 16, 3] where the expectation $\langle \cdot \rangle_{P_\theta(h|v) P^*(v)}$ corresponds to the E step and the gradient of $\log P_\theta(v, h)$ is related to the M step[1].

Instead of calculating analytically the true expectation in Eq. (2), it is possible to approximate it by sampling the visible sequences $v$ from the target distribution $P^*(v)$ and the hidden sequences from the posterior distribution $P_\theta(h|v)$ given the visible ones. Note that the posterior distribution $P_\theta(h|v)$ could be hard to sample from. Indeed, at a time $t$ the posterior distribution over $h_t$ does not only depend on the past visible activity but also on the future visible activity, since it is conditioned on the whole visible activity $v_{0:T}$ from time step 0 to $T$. This renders a true challenge for online algorithms. In the case of Hidden Markov Model training, the forward-backward algorithm

[4, 19] combines information from the past (by forward filtering) and from the future (by backward smoothing) to calculate $P_\theta(h|v)$.

If the statistical model does not have the Markovian property, the problem of calculating $P_\theta(h|v)$ (or sampling from it) becomes even harder. Here, we propose an alternative solution that does not require to sample from $P_\theta(h|v)$ and does not require the Markovian assumption (see [11, 17] for other approaches on sampling $P_\theta(h|v)$). We exploit that in all neuronal network models of interest, neuronal firing at any time point is conditionally independent given the past activity of the network. Using the chain rule this means that we can write the joint distribution $P_\theta(v, h)$ (see Fig. 1A) as

$$P_\theta(v,h) = \underbrace{\left( P_\theta(v_0) \prod_{t=1}^{T} \prod_{i=1}^{N_{\mathrm{v}}} P_\theta(v_{t,i}|v_{0:t-1}, h_{0:t-1}) \right)}_{R_\theta(v|h)} \underbrace{\left( P_\theta(h_0) \prod_{t=1}^{T} \prod_{i=1}^{N_{\mathrm{h}}} P_\theta(h_{t,i}|v_{0:t-1}, h_{0:t-1}) \right)}_{Q_\theta(h|v)},$$

(3)

where $R_\theta(v|h)$ is easy to calculate (see below) and $Q_\theta(h|v)$ is easy to sample from. The sampling can be accomplished by clamping the visible neurons to a target sequence $v$ and let the hidden dynamics run, i.e. at time $t$, $h_t$ is sampled from $P_\theta(h_t|v_{0:t-1}h_{0:t-1})$. [2] From Eq. (3), the posterior distribution $P_\theta(h|v)$ can be written as

$$P_\theta(h|v) = \frac{R_\theta(v|h)Q_\theta(h|v)}{P_\theta(v)},$$

(4)

where the marginal distribution over the visible sequences $v$ can be also expressed as $P_\theta(v) = \langle R_\theta(v|h) \rangle_{Q_\theta(h|v)}$. As a consequence, by using Eq. (4), the learning rule in Eq. (2) can be rewritten as

$$\Delta\theta = \sum_{v,h} P^*(v)P_\theta(h|v)\frac{\partial \log P_\theta(v,h)}{\partial\theta} = \sum_{v,h} P^*(v)Q_\theta(h|v)\frac{R_\theta(v|h)}{P_\theta(v)}\frac{\partial \log P_\theta(v,h)}{\partial\theta}$$

$$= \eta \left\langle \frac{R_\theta(v|h)}{\langle R_\theta(v|h') \rangle_{Q_\theta(h'|v)}} \frac{\partial \log P_\theta(v,h)}{\partial\theta} \right\rangle_{Q_\theta(h|v)P^*(v)}.$$

(5)

Instead of calculating the true expectation, Eq. (5) can be evaluated by using $N$ samples (see algorithm 1) where the factor $\gamma_\theta(v, h) := R_\theta(v|h)/\langle R_\theta(v|h') \rangle_{Q_\theta(h'|v)}$ acts as the importance weight [15]. Note that in the absence of hidden neurons, this factor $\gamma_\theta(v, h)$ is equal to one and the maximum likelihood learning rule [2, 18] is recovered.

**Algorithm 1** Sequence learning (batch mode)

---

Set an initial $\theta$
**while** $\theta$ not converged **do**
    $v \sim P^*(v)$
    $\alpha(v) = 0, P_\theta(v) = 0$
    **for** $i = 1 \ldots N$ **do**
        $h \sim Q_\theta(h|v)$
        $\alpha(v) \leftarrow \alpha(v) + R_\theta(v|h)\frac{\partial \log P_\theta(v,h)}{\partial\theta}$
        $P_\theta(v) \leftarrow P_\theta(v) + N^{-1}R_\theta(v|h)$
    **end for**
    $\theta \leftarrow \theta + \eta\frac{\alpha(v)}{P_\theta(v)}$
**end while**
**return** $\theta$

---

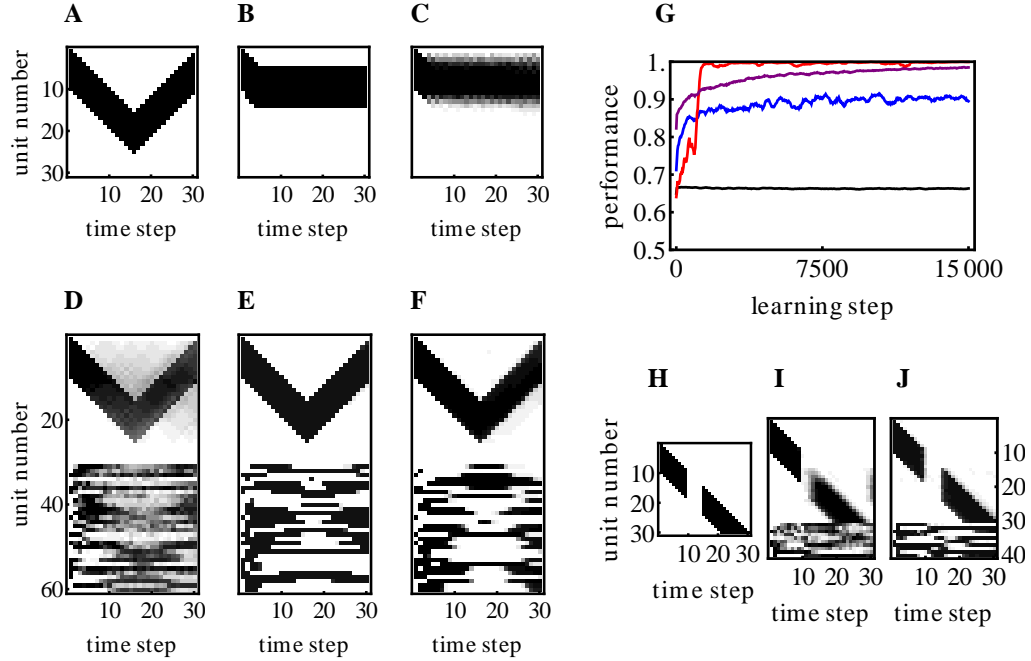

Figure 2: Learning a non-Markovian sequence of temporally correlated and linearly dependent states with different learning rules. **A** The target distribution contained only this training pattern for 30 visible neurons and 30 time steps. **B-F**, **H-J** Overlay of 20 recalls after learning with 15 000 training pattern presentations, **B** with only visible neurons and a simple asymmetric Hebb rule (see main text) **C** only visible neurons and learning rule Eq. (5) **D** static weights towards 30 hidden neurons (Reservoir Computing) **E** learning rule Eq. (5), **F** online approximation Eq. (14). **G** Learning curves for the training pattern in A for only visible neurons (black line), static weights towards hidden (blue line), online learning approximation (purple line) exact learning rule (red line). The performance was measured in one minus average Hamming distance per neuron per time step (see main text). **H** A training pattern that exhibits a gap of 5 time-steps. **I** Recall with a network of 30 visible and 10 hidden neurons without learning the weights towards hidden neurons. **J** Recall after training the same network with learning rule Eq. (5).

## 3 Binary neurons

In order to illustrate the learning rule given by Eq. (5), let us consider sequences of binary patterns. Let $x$ denote the activity of the visible and hidden neurons, i.e. $x = (v, h)$. Since the individual neurons are binary $x_{t,i} \in \{-1, 1\}$, their distribution is given by $P_\theta(x_{t,i}|x_{0:t-1}) = (\rho_{t,i}\delta t)^{(1+x_{t,i})/2}(1 - \rho_{t,i}\delta t)^{(1-x_{t,i})/2}$, where the firing rate $\rho_{t,i}$ of neuron $i$ at time $t$ is given by a monotonically increasing (and non-linear) function $g$ of its membrane potential $u_{t,i}$, i.e.

$$\rho_{t,i} = g(u_{t,i}) \quad \text{with } u_{t,i} = \sum_j w_{ij} x_{t-1,j}. \tag{6}$$

Note that these assumptions lead to Markovian neuronal dynamics i.e. $P_\theta(x_{t,i}|x_{0:t-1}) = P_\theta(x_{t,i}|x_{t-1})$ (see Fig. 1B). Further calculations will be slightly simplified, if we assume that the non-linear function $g$ is constraint by the following differential equation $dg(u)/du = \beta g(u)(1 - g(u)\delta t)$. Note that in the limit of $\delta t \to 0$, this function is an exponential, i.e. $g(u) = g_0 \exp(\beta u)$ and for finite $\delta t$, it is a sigmoidal and takes the form $g(u) = \delta t^{-1} \left(1 + \left((g_0\delta t)^{-1} - 1\right) \exp(-\beta u)\right)^{-1}$, where we constrained the solutions such that $g(0) = g_0$ in order to be consistent with the case where $\delta t \to 0$.

For the distribution over the initial conditions $P_\theta(v_0)$ and $P_\theta(h_0)$ we choose delta distributions such that $v_0$ is equal to the first state of the training sequence and $h_0$ is an arbitrary but fixed vector of binary values. If we assume that the weights $w_{ij}$ are the only adaptable parameters in this model,

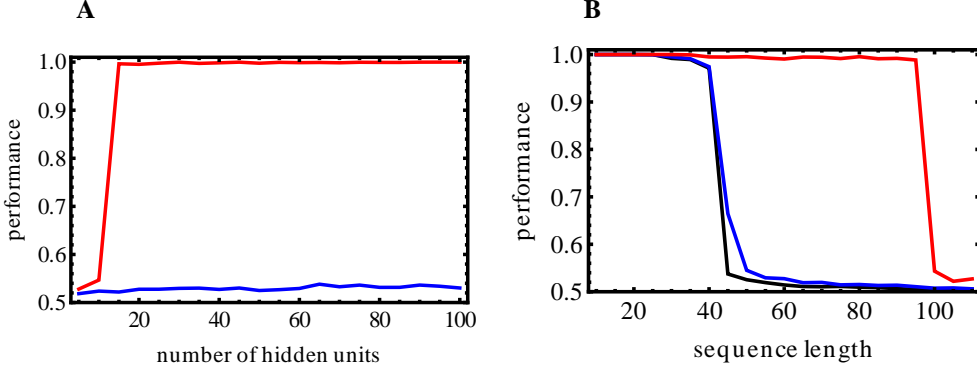

Figure 3: Adding trainable hidden neurons leads to much better recall performance than having static hidden neurons or no hidden neurons at all. **A** Comparison of the performance after 20000 learning cycles between static (blue curve) and dynamic weights (red curve) towards hidden neurons for a network with 30 visible and different numbers of hidden neurons in a training task with a uncorrelated random pattern of length 60 time steps. For **B** we generated random, uncorrelated sequences of different length and compared the performance after 20000 learning cycles for only visible neurons (black curve), static weights towards hidden (blue curve) and dynamic weights towards hidden (red curve).

we have

$$\frac{\partial \log P_w(x_{t,i}|x_{0:t-1})}{\partial w_{ij}} = \frac{1}{2}\left((1+x_{t,i})\frac{g'(u_{t,i})}{g(u_{t,i})} - (1-x_{t,i})\frac{g'(u_{t,i})\delta t}{1-g(u_{t,i})\delta t}\right)\frac{\partial u_{t,i}}{\partial w_{ij}}. \tag{7}$$

With the above assumption on $g(u)$ and Eq. (3) and (6) we find

$$\frac{\partial \log P_w(x)}{\partial w_{ij}} = \frac{\beta}{2}\sum_{t=1}^{T}(x_{t,i} - \langle x_{t,i}\rangle_{P_\theta(x_{t,i}|x_{t-1})})x_{t-1,j}, \tag{8}$$

where $\langle x_{t,i}\rangle_{P_\theta(x_{t,i}|x_{t-1})} = g(u_{t,i})\delta t - (1 - g(u_{t,i})\delta t)$ and the indices $i$ and $j$ run over all visible and hidden neurons. The factor $R_w(v|h)$ can be expressed as

$$R_w(v|h) = \exp\left(\frac{1}{2}\sum_{t=0}^{T}\sum_{i=1}^{N_v}(1+v_{t,i})\log(\rho_{t,i}\delta t) + (1-v_{t,i})\log(1-\rho_{t,i}\delta t)\right). \tag{9}$$

Let us now consider a simple case (Fig. 2) where the distribution over sequences is a delta distribution $P^*(v) = \delta(v - v^*)$ around a single pattern $v^*$ (Fig. 2A) which is made of a set of temporally correlated and linearly dependent states $\{v_t^*\}_{t=0}^{T}$, i.e. a non-Markovian pattern, thus making it a difficult pattern to learn with a simple asymmetric Hebb rule $\Delta w_{ij} \propto \sum_{t=0}^{T} v_{t+1,i}^* v_{t,j}^*$ (Fig. 2B) or only visible neurons (Fig. 2C), which are both Markovian learning rules. The performance was measured by one minus the Hamming distance per visible neuron and time step $1-(TN_v)^{-1}\sum_{t,i}|v_{t,i}-v_{t,i}^*|/2$ between target pattern and recall pattern averaged over 100 runs. Adding hidden neurons without learning the weights towards hidden neurons is similar to the idea used in the framework of Reservoir Computing (for a review see [13]): the visible states feed a fixed reservoir of neurons that returns a non-linear transformation of the input. Only the readout from hidden to visible neurons and in our case the recurrent connections in the visible layer are trained. To assure a sensible distribution of weights towards hidden units, we used the weights that were obtained after learning with Eq. (5) and reshuffled them. Obviously, without training the reservoir the performance is always worse compared to a system with an equal number of hidden neurons but dynamic weights (Fig. 2E and 2F). With only a few hidden neurons our rule is also capable to learn patterns where the visible neurons are silent during a few time-steps. The training pattern in Fig. 2H exhibits a gap of 5 time steps. After learning the weights towards 10 hidden neurons with learning rule Eq. (5) recall performance is nearly perfect (see Fig. 2J). With only visible neurons (not shown in Fig. 2) or static weights towards hidden neurons the time gap was not learned (see Fig. 2I).

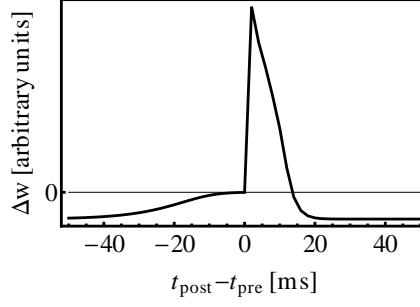

Figure 4: The learning rule Eq. (11) is compatible with Spike-Timing Dependent Plasticity (STDP): the weight gets potentiated if a presynaptic spike is followed by a postsynaptic spike and depressed otherwise. The time course of the postsynaptic potential and the refractory kernel is given in the text.

In Fig. 3 we used again delta target distributions $P^*(v) = \delta(v - v^*)$ with random uncorrelated patterns $v^*$ of different length. Each model was trained with 20000 pattern presentations. For a pattern of length $2N_v = 60$ only $N_v/2 = 15$ trainable hidden neurons are sufficient to reach perfect recall (see Fig. 3A). This is in clear contrast to the case of static hidden weights. Again the static weights were obtained by reshuffling those that we obtained after learning with Eq. (5). Fig. 3B compares the capacity of our learning rule with $N_h = N_v = 30$ hidden neurons to the case of no hidden neuron or static weights towards hidden neurons. Without learning the weights towards hidden neurons the performance drops to almost chance level for sequences of 45 or more time steps, whereas with our learning rule this decrease of performance occurs only at sequences of 100 or more time steps.

## 4    Limit to Continuous Time

Starting from the neurons in the last section we show that in the limit to continuous time we can implement the sequence learning task with stochastic spiking neurons [7].

First note that the state of a neuron at time $t$ in the model described in the previous section is fully defined by $u_{t,i} := \sum_j w_{ij} x_{t-1,j}$ (see Eq. (6)) and its spiking activity $x_{t,i}$. The weighted sum $\sum_j w_{ij} x_{t-1,j}$ is the response of neuron $i$ to the spikes of its presynaptic neurons and its own spikes. The terms in this sum depend on the previous time step only. In a more realistic model the postsynaptic neuron feels the influence of presynaptic spikes through a perturbation of the membrane potential on the order of a few milliseconds, which in the limit to continuous time clearly cannot be modeled by a one-time step response. For a more realistic model we replace $u_{t,i}$ in Eq. (6) by

$$u_{t,i} = \underbrace{\sum_{s=1}^{\infty} \kappa_s x_{t-s,i}}_{=:x_{t,i}^{\kappa}} + \sum_{j \neq i} w_{ij} \underbrace{\sum_{s=1}^{\infty} \epsilon_s x_{t-s,j}}_{=:x_{t,j}^{\epsilon}}, \qquad (10)$$

where $x_{t-s,i} \in \{0,1\}$. The kernel $\epsilon$ models the time-course of the response to a presynaptic spike and $\kappa$ the refractoriness. Our model holds for any choices of $\epsilon$ and $\kappa$, including for example a hard refractory period where the neuron is forced not to spike.

In order to take the limit $\delta t \to 0$ in Eq. (9) we note that we can scale $R_w(v|h)$ without changing the learning rule Eq. (5), since there only the ratio $R_\theta(v|h)/\langle R_\theta(v|h') \rangle_{Q_\theta(h'|v)}$ enters. We use the scaling $R_w(v|h) \to \widetilde{R}_w(v|h) := (g_0 \delta t)^{-S_v} R_w(v|h)$, where $S_v$ denotes the total number of spikes in the visible sequence $v$, i.e. $S_v = \sum_{t=0}^{T} \sum_{i=1}^{N_v} v_{t,i}$. Note that for $(0,1)$-units the expectation in Eq. (8) becomes $\langle x_{t,i} \rangle_{P_\theta(x_{t,i}|x_{t-1})} = g(u_{t,i}) \delta t = \rho_{t,i} \delta t$ . Now we take the limit $\delta t \to 0$ in Eq. (8)

and (9) and find

$$\frac{\partial \log P_w(x)}{\partial w_{ij}} = \int_0^T dt \, \beta(x_i(t) - \rho_i(t)) x_j^\epsilon(t) \tag{11}$$

$$\widetilde{R}_w(v|h) = \exp\left(\int_0^T dt \sum_{i=1}^{N_v} \beta v_i(t) u_i(t) - \rho_i(t)\right), \tag{12}$$

where the training pattern runs from time 0 to $T$, $x_i(t) = \sum_{t_i^{(f)}} \delta(t - t_i^{(f)})$ is the sum of delta spikes of neuron $i$ at times $t_i^{(f)}$, $x_j^\epsilon(t) = \int ds \, \epsilon(s) x_j(t-s)$ (and similarly $x_i^\kappa(t)$) is the convolution of presynaptic spike trains with the response kernel $\epsilon(t)$. With neuron $i$'s response to past spiking activity $u_i(t) = x_i^\kappa(t) + \sum_{j \neq i} w_{ij} x_j^\epsilon(t)$ and the escape rate function $\rho_i(t) = g_0 \exp(\beta u_i(t))$ we recovered the defining equations of a simplified stochastic spike response model [7].

In Fig. 4 we display the weight change after forcing two neurons to fire with a fixed time lag. For the figure we used the kernels $\epsilon_s \propto \exp(-s/\tau_m) - \exp(-s/\tau_s)$ and $\kappa_s \propto -\exp(-s/\tau_m)$ with $\tau_m = 10$ ms and $\tau_s = 2$ ms. Our learning rule is consistent with STDP in the sense that a presynaptic spike followed by a postsynaptic spike leads to potentiation and to depression otherwise. Note that this result was also found in [18].

## 5  Approximate online version

Without hidden neurons the learning rule found by using Eq. (11) is straightforward to implement in an online way where the parameters are updated at every moment in time according to $\dot{w}_{ij} \propto (x_i(t) - \rho_i(t)) x_j^\epsilon(t)$ instead of waiting with the update until a training batch finished. Finding an online version of the learning algorithm for networks with hidden neurons turns out to be a challenge, since we need to know the whole sequences $v$ and $h$ in order to evaluate the importance factor $R_\theta(v|h)/\langle R_\theta(v|h') \rangle_{Q_\theta(h'|v)}$. Here we propose to use in each time step an approximation of the importance factor based on the network dynamics during the preceding period of typical sequence length and multiply it by the low-pass filtered change of parameters. We write this section with $x_i(t) \in \{0, 1\}$, but similar expressions are easily found for $x_i(t) \in \{-1, 1\}$.

---
**Algorithm 2** Sequence learning (online mode)
***

Set an initial $w_{ij}, e_{ij}, a, \bar{r}, t$
**while** $w_{ij}$ not converged **do**
  **if** $t \mod NT == 0$ **then**
    $v \sim P^*(v)$
  **end if**
  $s = t \mod T$
  **if** $s < \tau$ **then**
    $h(s) \sim P(h(s))$ **else** $h(s) \sim P_w(h(s)|\text{past spiking activity})$
  **end if**
  $x(s) = (v(s), h(s))$
  $e_{ij} \leftarrow (1 - \frac{\delta t}{T}) e_{ij} + \beta(x_i(s) - \rho_i(s)) x_j^\epsilon(s)$
  $a \leftarrow (1 - \frac{\delta t}{T}) a + \sum_{i=1}^{N_v} \beta v_i(s) u_i(s) - \rho_i(s)$
  $\bar{r} \leftarrow (1 - \frac{\delta t}{NT}) \bar{r} + \exp(a)$
  $w_{ij} \leftarrow w_{ij} + \eta \frac{\exp(a)}{\bar{r}} e_{ij}$
  $t \leftarrow t + \delta t$
**end while**
**return** $w_{ij}$

---

In Eq. (13a) and (13b) we summarize how to use low-pass filters to approximate the integrals in Eq. (11) and Eq. (12). The time constant of the low-pass filter is chosen to match the sequence length $T$. To find an online estimate of $\langle R_\theta(v, h') \rangle_{Q_\theta(h'|v)}$ we assume that a training pattern $v \sim P^*(v)$ is presented a few times in a row and after time $NT$, with $N \in \mathbb{N}, N \gg 1$, a new training pattern is picked from the training distribution. Under this assumption we can replace the average over

hidden sequences by a low-pass filter of $r$ with time constant $NT$, see Eq. (13c). At the beginning of each pattern presentation - i.e. during the time interval $[0, \tau)$, with $\tau$ on the order of the kernel time constant $\tau_m$ - the hidden activity $h(s)$ is drawn from a given distribution $P(h(s))$.

$$\dot{e}_{ij}(t) = -\frac{1}{T}e_{ij}(t) + \beta(x_i(t) - \rho_i(t))x_j^{\epsilon}(t) \qquad e_{ij}(T) \approx \frac{\partial \log P_w(x)}{\partial w_{ij}} \qquad (13a)$$

$$\dot{a}(t) = -\frac{1}{T}a(t) + \sum_{i=1}^{N_{\mathrm{v}}} \beta v_i(t)u_i(t) - \rho_i(t) \qquad \exp(a(T)) \approx R_w(v|h) \qquad (13b)$$

$$NT\dot{\bar{r}}(t) = -\bar{r}(t) + r(t), \qquad r(t) := \exp(a(t)) \qquad \bar{r}(NT) \approx \langle R_\theta(v, h') \rangle_{Q_\theta(h'|v)} \qquad (13c)$$

Finally we learn the model parameters in each time step according to

$$\dot{w}_{ij}(t) = \eta \frac{r(t)}{\bar{r}(t)} e_{ij}(t) . \qquad (14)$$

This online algorithm is certainly a rough approximation of the batch algorithm. Nevertheless, when applied to the challenging example (Fig. 2A) in section 3, the performance of the online rule is close to the one of the batch rule (Fig. 2F, G).

## 6 Discussion

Learning long and temporally correlated sequences with neural networks is a difficult task. In this paper we suggested a statistical model with hidden neurons and derived a learning rule that leads to optimal recall of the learned sequences given the neuronal dynamics. The learning rule is derived by minimizing the Kullback-Leibler divergence from training distribution to model distribution with a variant of the EM-algorithm, where we use importance sampling to draw hidden sequences given the visible training sequence. Choosing an appropriate distribution in the importance sampling step we are able to circumvent inference which usually makes the training of non-Markovian models hard. The resulting learning algorithm consists of a local term modulated by a global factor. We showed that it is ready to be implemented with biologically realistic neurons and that an approximate online version exists.

Our approach follows the ideas outlined in [2], where sequence learning was considered with visible neurons. Here we extended this model by adding stochastic hidden neurons that help to perform well with sequences of linearly depend states - including non-Markovian sequences - or long sequences. As in [18] we look at the limit of continuous time and find that the learning rule is consistent with Spike-Timing Dependent Plasticity. In contrast to Reservoir Computing [13] we train the weights towards hidden neurons which clearly helps to improve performance. Our learning rule does not need a "wake" and a "sleep" phase as we know it from Boltzmann machines [1, 8].

Viewed in a different light our learning algorithm has a nice interpretation: as in reinforcement learning, the hidden neurons explore different sequences, where each trial leads to a global reward signal that modulates the weight change. However, in contrast to common reinforcement learning the reward is not provided by an external teacher but depends solely on the internal dynamics and the visible neurons do not explore but are clamped to the training sequence.

To make our model even more biologically relevant, future work should aim for a biological implementation of the global importance factor that depends on the spike timing and the membrane potential of all the visible neurons (see Eq. (9)). It would also be interesting to study online approximations of the learning algorithm in more detail or its application to models with the Hidden Markov structure.

### Acknowledgments

The authors thank Robert Urbanczik for helpful discussions. This work was supported by the Swiss National Science Foundation (SNF), grant 31-133094, and a grant from the Swiss SystemsX.ch initiative (Neurochoice, evaluated by the SNF).

## Footnotes

[1]Strictly speaking the M step of the EM algorithm directly calculates the solution $\theta^{\mathrm{new}}$ for which $\frac{\partial}{\partial \theta} \log P_\theta(v, h) = 0$ whereas in Eq. (2) there is only one step done in the direction of the gradient.

[2]Note that for other conditional dependencies it might be reasonable to split $P_\theta(h|v)$ differently. For example in models with the structure of Hidden Markov Models one could make use of the fact that $P_\theta(h|v) = \prod_{t=0}^{T-1} P_\theta(h_t|v_{0:t}, h_{t+1}) = \prod_{t=0}^{T-1} \frac{P_\theta(h_{t+1}|h_t)}{P_\theta(h_{t+1}|v_{0:t})} P_\theta(h_t|v_{0:t})$ and take the product of filtering distributions $Q_\theta(h|v) = \prod_{t=0}^{T-1} P_\theta(h_t|v_{0:t})$ to sample from and use the importance weights $R_\theta(v, h) = \prod_{t=0}^{T-1} \frac{P_\theta(h_{t+1}|h_t)}{P_\theta(h_{t+1}|v_{0:t})}$. Following the reasoning in the main text one finds an alternative to the forward-backward algorithm [4, 19] that might be interesting to investigate further.

## References

[1] D. Ackley and G. E. Hinton. A learning algorithm for boltzmann machines. *Cognitive Science*, 9(1):147–169, 1985.

[2] D. Barber. Learning in spiking neural assemblies. *Advances in Neural Information Processing Systems*, 15, 2003.

[3] D. Barber. *Bayesian Reasoning and Machine Learning*. Cambridge University Press, 2011. In press.

[4] L. Baum, T. Petrie, G. Soules, and N. Weiss. A maximization technique occurring in the statistical analysis of probabilistic functions of Markov chains. *The Annals of Mathematical Statistics*, 41(1):164–171, 1970.

[5] A. Dempster, N. Laird, and D. Rubin. Maximum likelihood from incomplete data via the EM algorithm. *Journal of the Royal Statistical Society. Series B (Methodological)*, 39(1):1–38, 1977.

[6] A. Düring, A. Coolen, and D. Sherrington. Phase diagram and storage capacity of sequence processing neural networks. *Journal of Physics A: Mathematical and General*, 31:8607, 1998.

[7] W. Gerstner and W. M. Kistler. *Spiking neuron models: single neurons, populations, plasticity*. Cambridge University Press, 2002.

[8] G. E. Hinton. Training products of experts by minimizing contrastive divergence. *Neural Computation*, 14(8):1771–800, 2002.

[9] G. E. Hinton and A. Brown. Spiking boltzmann machines. *Advances in Neural Information Processing Systems*, 12, 2000.

[10] J. Hopfield. Neural networks and physical systems with emergent collective computational abilities. *Proceedings of the National Academy of Sciences of the United States of America*, 79(8):2554, 1982.

[11] P. Latham and J. W. Pillow. Neural characterization in partially observed populations of spiking neurons. *Advances in Neural Information Processing Systems*, 20:1161–1168, 2008.

[12] M. Lengyel, J. Kwag, O. Paulsen, and P. Dayan. Matching storage and recall: hippocampal spike timing-dependent plasticity and phase response curves. *Nature Neuroscience*, 8(12):1677–83, 2005.

[13] M. Lukoševičius and H. Jaeger. Reservoir computing approaches to recurrent neural network training. *Computer Science Review*, 3(3):127–149, 2009.

[14] W. Maass, T. Natschläger, and H. Markram. Real-time computing without stable states: a new framework for neural computation based on perturbations. *Neural Computation*, 14(11):2531–60, 2002.

[15] D. J. C. MacKay. *Information Theory, Inference & Learning Algorithms*. Cambridge University Press, 2002.

[16] G. McLachlan and T. Krishnan. *The EM Algorithm and Extensions*. John Wiley and Sons, 1997.

[17] Y. Mishchenko and L. Paninski. Efficient methods for sampling spike trains in networks of coupled neurons. *The Annals of Applied Statistics*, 5(3):1893–1919, 2011.

[18] J.-P. Pfister, T. Toyoizumi, D. Barber, and W. Gerstner. Optimal spike-timing-dependent plasticity for precise action potential firing in supervised learning. *Neural Computation*, 18(6):1318–1348, 2006.

[19] L. Rabiner. A tutorial on hidden Markov models and selected applications in speech recognition. *Proceedings of the IEEE*, 77(2):257–86, 1989.

[20] H. Sompolinsky and I. Kanter. Temporal association in asymmetric neural networks. *Physical Review Letters*, 57(22):2861–64, 1986.

[21] I. Sutskever, G. E. Hinton, and G. Taylor. The Recurrent Temporal Restricted Boltzmann Machine. *Advances in Neural Information Processing Systems*, 21:1601–08, 2009.

[22] G. Taylor, G. E. Hinton, and S. Roweis. Modeling human motion using binary latent variables. *Advances in Neural Information Processing Systems*, 19:1345–52, 2007.

